# Sample Complexity for Learning Recurrent Perceptron Mappings

**Bhaskar Dasgupta**
Department of Computer Science
University of Waterloo
Waterloo, Ontario N2L 3G1
CANADA
bdasgupt@daisy.uwaterloo.ca

**Eduardo D. Sontag**
Department of Mathematics
Rutgers University
New Brunswick, NJ 08903
USA
sontag@control.rutgers.edu

## Abstract

Recurrent perceptron classifiers generalize the classical perceptron model. They take into account those correlations and dependences among input coordinates which arise from linear digital filtering. This paper provides tight bounds on sample complexity associated to the fitting of such models to experimental data.

## 1 Introduction

One of the most popular approaches to binary pattern classification, underlying many statistical techniques, is based on *perceptrons* or *linear discriminants*; see for instance the classical reference (Duda and Hart, 1973). In this context, one is interested in classifying $k$-dimensional input patterns

$$v = (v_1, \ldots, v_k)$$

into two disjoint classes $A^+$ and $A^-$. A perceptron $P$ which classifies vectors into $A^+$ and $A^-$ is characterized by a vector (of "weights") $\vec{c} \in \mathbb{R}^k$, and operates as follows. One forms the inner product

$$\vec{c}.v = c_1 v_1 + \ldots c_k v_k.$$

If this inner product is positive, $v$ is classified into $A^+$, otherwise into $A^-$.

In signal processing and control applications, the size $k$ of the input vectors $v$ is typically very large, so the number of samples needed in order to accurately "learn" an appropriate classifying perceptron is in principle very large. On the other hand, in such applications the classes $A^+$ and $A^-$ often can be separated by means of a dynamical system of fairly small dimensionality. The existence of such a dynamical system reflects the fact that the signals of interest exhibit context dependence and

correlations, and this prior information can help in narrowing down the search for a classifier. Various dynamical system models for classification appear from instance when learning finite automata and languages (Giles *et. al.*, 1990) and in signal processing as a channel equalization problem (at least in the simplest 2-level case) when modeling linear channels transmitting digital data from a quantized source, e.g. (Baksho et. al., 1991) and (Pulford *et. al.*, 1991).

When dealing with linear dynamical classifiers, the inner product $\vec{c}.v$ represents a convolution by a separating vector $\vec{c}$ that is the impulse-response of a recursive digital filter of some order $n \ll k$. Equivalently, one assumes that the data can be classified using a $\vec{c}$ that is *n-recursive*, meaning that there exist real numbers $r_1, \ldots, r_n$ so that

$$c_j = \sum_{i=1}^{n} c_{j-i} r_i, \quad j = n+1, \ldots, k.$$

Seen in this context, the usual perceptrons are nothing more than the very special subclass of "finite impulse response" systems (all poles at zero); thus it is appropriate to call the more general class "recurrent" or "IIR (infinite impulse response)" perceptrons. Some authors, particularly Back and Tsoi (Back and Tsoi, 1991; Back and Tsoi, 1995) have introduced these ideas in the neural network literature. There is also related work in control theory dealing with such classifying, or more generally quantized-output, linear systems; see (Delchamps, 1989; Koplon and Sontag, 1993).

The problem that we consider in this paper is: if one assumes that there is an *n*-recursive vector $\vec{c}$ that serves to classify the data, and one knows $n$ but not the particular vector, how many labeled samples $v^{(i)}$ are needed so as to be able to reliably estimate $\vec{c}$? More specifically, we want to be able to guarantee that any classifying vector consistent with the seen data will classify "correctly with high probability" the unseen data as well. This is done by computing the VC dimension of the related concept class and then applying well-known results from computational learning theory. Very roughly speaking, the main result is that the number of samples needed is proportional to the logarithm of the length $k$ (as opposed to $k$ itself, as would be the case if one did not take advantage of the recurrent structure). Another application of our results, again by appealing to the literature from computational learning theory, is to the case of "noisy" measurements or more generally data not exactly classifiable in this way; for example, our estimates show roughly that if one succeeds in classifying 95% of a data set of size $\log q$, then with confidence $\approx 1$ one is assured that the prediction error rate will be < 90% on future (unlabeled) samples.

Section 5 contains a result on polynomial-time learnability: for $n$ constant, the class of concepts introduced here is PAC learnable. Generalizations to the learning of real-valued (as opposed to Boolean) functions are discussed in Section 6. For reasons of space we omit many proofs; the complete paper is available by electronic mail from the authors.

## 2   Definitions and Statements of Main Results

Given a set $\mathbb{X}$, and a subset $X$ of $\mathbb{X}$, a *dichotomy* on $X$ is a function

$$\delta : X \to \{-1, 1\}.$$

Assume given a class $\mathcal{F}$ of functions $\mathbb{X} \to \{-1, 1\}$, to be called the class of *classifier* functions. The subset $X \subseteq \mathbb{X}$ is *shattered* by $\mathcal{F}$ if each dichotomy on $X$ is the restriction to $X$ of some $\phi \in \mathcal{F}$. The *Vapnik-Chervonenkis dimension* VC $(\mathcal{F})$ is the supremum (possibly infinite) of the set of integers $\kappa$ for which there is some subset

$X \subseteq \mathbb{X}$ of cardinality $\kappa$ which can be shattered by $\mathcal{F}$. Due to space limitations, we omit any discussion regarding the relevance of the VC dimension to learning problems; the reader is referred to the excellent surveys in (Maass, 1994; Turán, 1994) regarding this issue.

Pick any two integers $n > 0$ and $q \geq 0$. A sequence

$$\vec{c} = (c_1, \dots, c_{n+q}) \in \mathbb{R}^{n+q}$$

is said to be *n-recursive* if there exist real numbers $r_1, \dots, r_n$ so that

$$c_{n+j} = \sum_{i=1}^{n} c_{n+j-i} r_i, \quad j = 1, \dots, q.$$

(In particular, every sequence of length $n$ is $n$-recursive, but the interesting cases are those in which $q \neq 0$, and in fact $q \gg n$.) Given such an $n$-recursive sequence $\vec{c}$, we may consider its associated *perceptron* classifier. This is the map

$$\phi_{\vec{c}} : \mathbb{R}^{n+q} \to \{-1, 1\} : \quad (x_1, \dots, x_{n+q}) \mapsto \mathrm{sign}\left(\sum_{i=1}^{n+q} c_i x_i\right)$$

where the sign function is understood to be defined by $\mathrm{sign}(z) = -1$ if $z \leq 0$ and $\mathrm{sign}(z) = 1$ otherwise. (Changing the definition at zero to be $+1$ would not change the results to be presented in any way.) We now introduce, for each two fixed $n, q$ as above, a class of functions:

$$\mathcal{F}_{n,q} := \left\{ \phi_{\vec{c}} \mid \vec{c} \in \mathbb{R}^{n+q} \text{ is } n\text{-recursive} \right\}.$$

This is understood as a function class with respect to the input space $\mathbb{X} = \mathbb{R}^{n+q}$, and we are interested in estimating $\mathrm{vc}(\mathcal{F}_{n,q})$.

Our main result will be as follows (all logs in base 2):

**Theorem 1**

$$\boxed{\max\left\{n, n\lfloor \log(\lfloor 1 + \tfrac{q-1}{n}\rfloor)\rfloor\right\} \leq \mathrm{vc}(\mathcal{F}_{n,q}) \leq \min\{n+q, 18n + 4n\log(q+1)\}}$$

Note that, in particular, when $q > \max\{2 + n^2, 32\}$, one has the tight estimates

$$\frac{n}{2}\log q \leq \mathrm{vc}(\mathcal{F}_{n,q}) \leq 8n \log q.$$

The organization of the rest of the paper is as follows. In Section 3 we state an abstract result on VC-dimension, which is then used in Section 4 to prove Theorem 1. Finally, Section 6 deals with bounds on the sample complexity needed for identification of linear dynamical systems, that is to say, the real-valued functions obtained when not taking "signs" when defining the maps $\phi_{\vec{c}}$.

## 3   An Abstract Result on VC Dimension

Assume that we are given two sets $\mathbb{X}$ and $\Lambda$, to be called in this context the set of *inputs* and the set of *parameter values* respectively. Suppose that we are also given a function

$$F : \Lambda \times \mathbb{X} \to \{-1, 1\}.$$

Associated to this data is the class of functions

$$\mathcal{F} := \{F(\lambda, \cdot) : \mathbb{X} \to \{-1, 1\} \mid \lambda \in \Lambda\}$$

obtained by considering $F$ as a function of the inputs alone, one such function for each possible parameter value $\lambda$. Note that, given the same data one could, dually, study the class

$$\mathcal{F}^* \ : \ \{F(\cdot, \xi) : \Lambda \rightarrow \{-1, 1\} \, | \, \xi \in \mathbb{X}\}$$

which obtains by fixing the elements of $\mathbb{X}$ and thinking of the parameters as inputs. It is well-known (and in any case, a consequence of the more general result to be presented below) that $\mathrm{vc}\,(\mathcal{F}) \geq \lfloor \log(\mathrm{vc}\,(\mathcal{F}^*)) \rfloor$, which provides a lower bound on $\mathrm{vc}\,(\mathcal{F})$ in terms of the "dual VC dimension." A sharper estimate is possible when $\Lambda$ can be written as a product of $n$ sets

$$\Lambda = \Lambda_1 \times \Lambda_2 \times \ldots \times \Lambda_n \tag{1}$$

and that is the topic which we develop next.

We assume from now on that a decomposition of the form in Equation (1) is given, and will define a variation of the dual VC dimension by asking that only certain dichotomies on $\Lambda$ be obtained from $\mathcal{F}^*$. We define these dichotomies only on "rectangular" subsets of $\Lambda$, that is, sets of the form

$$L = L_1 \times \ldots \times L_n \subseteq \Lambda$$

with each $L_i \subseteq \Lambda_i$ a nonempty subset. Given any index $1 \leq \kappa \leq n$, by a $\kappa$-*axis dichotomy* on such a subset $L$ we mean any function $\delta : L \rightarrow \{-1, 1\}$ which depends only on the $\kappa$th coordinate, that is, there is some function $\phi : L_\kappa \rightarrow \{-1, 1\}$ so that $\delta(\lambda_1, \ldots, \lambda_n) = \phi(\lambda_\kappa)$ for all $(\lambda_1, \ldots, \lambda_n) \in L$; an axis dichotomy is a map that is a $\kappa$-axis dichotomy for some $\kappa$. A rectangular set $L$ will be said to be *axis-shattered* if every axis dichotomy is the restriction to $L$ of some function of the form $F(\cdot, \xi) : \Lambda \rightarrow \{-1, 1\}$, for some $\xi \in \mathbb{X}$.

**Theorem 2** *If $L = L_1 \times \ldots \times L_n \subseteq \Lambda$ can be axis-shattered and each set $L_i$ has cardinality $r_i$, then $\mathrm{vc}\,(\mathcal{F}) \geq \lfloor \log(r_1) \rfloor + \ldots + \lfloor \log(r_n) \rfloor$.*

(In the special case $n=1$ one recovers the classical result $\mathrm{vc}\,(\mathcal{F}) \geq \lfloor \log(\mathrm{vc}\,(\mathcal{F}^*)) \rfloor$.) The proof of Theorem 2 is omitted due to space limitations.

## 4   Proof of Main Result

We recall the following result; it was proved, using Milnor-Warren bounds on the number of connected components of semi-algebraic sets, by Goldberg and Jerrum:

**Fact 4.1** (Goldberg and Jerrum, 1995) Assume given a function $F : \Lambda \times \mathbb{X} \rightarrow \{-1, 1\}$ and the associated class of functions $\mathcal{F} := \{F(\lambda, \cdot) : \mathbb{X} \rightarrow \{-1, 1\} \, | \, \lambda \in \Lambda\}$. Suppose that $\Lambda = \mathbb{R}^k$ and $\mathbb{X} = \mathbb{R}^n$, and that the function $F$ can be defined in terms of a Boolean formula involving at most $s$ polynomial inequalities in $k + n$ variables, each polynomial being of degree at most $d$. Then, $\mathrm{vc}\,(\mathcal{F}) \leq 2k \log(8eds)$.     $\square$

Using the above Fact and bounds for the standard "perceptron" model, it is not difficult to prove the following Lemma.

**Lemma 4.2** $\mathrm{vc}\,(\mathcal{F}_{n,q}) \leq \min\{n + q, \, 18n + 4n \log(q + 1)\}$

Next, we consider the lower bound of Theorem 1.

**Lemma 4.3** $\mathrm{vc}\,(\mathcal{F}_{n,q}) \geq \max\{n, \, n \lfloor \log(\lfloor 1 + \frac{q-1}{n} \rfloor) \rfloor\}$

*Proof.* As $\mathcal{F}_{n,q}$ contains the class of functions $\phi_{\vec{c}}$ with $\vec{c} = (c_1, \ldots, c_n, 0, \ldots, 0)$, which in turn being the set of signs of an $n$-dimensional linear space of functions, has VC dimension $n$, we know that $\text{VC}\,(\mathcal{F}_{n,q}) \geq n$. Thus we are left to prove that if $q > n$ then $\text{VC}\,(\mathcal{F}_{n,q}) \geq n\lfloor\log(\lfloor 1 + \frac{q-1}{n}\rfloor)\rfloor$.

The set of $n$-recursive sequences of length $n + q$ includes the set of sequences of the following special form:

$$c_j = \sum_{i=1}^{n} l_i^{j-1} \,, \quad j = 1, \ldots, n+q \tag{2}$$

where $\alpha_i, l_i \in \mathbb{R}$ for each $i = 1, \ldots, n$. Hence, to prove the lower bound, it is sufficient to study the class of functions induced by

$$F : \mathbb{R}^n \times \mathbb{R}^{n+q} \to \{-1, 1\}\,, \quad (\lambda_1, \ldots, \lambda_n, x_1, \ldots, x_{n+q}) \mapsto \text{sign}\left(\sum_{i=1}^{n}\sum_{j=1}^{n+q} \lambda_i^{j-1} x_j\right).$$

Let $r = \lfloor\frac{q+n-1}{n}\rfloor$ and let $L_1, \ldots, L_n$ be $n$ disjoint sets of real numbers (if desired, integers), each of cardinality $r$. Let $L = \bigcup_{i=1}^{n} L_i$. In addition, if $rn < q+n-1$, then select an additional set $B$ of $(q+n-rn-1)$ real numbers disjoint from $L$.

We will apply Theorem 2, showing that the rectangular subset $L_1 \times \ldots \times L_n$ can be axis-shattered. Pick any $\kappa \in \{1, \ldots, n\}$ and any $\phi : L_\kappa \to \{-1, 1\}$. Consider the (unique) interpolating polynomial

$$p(\lambda) = \sum_{j=1}^{n+q} x_j \lambda^{j-1}$$

in $\lambda$ of degree $q + n - 1$ such that

$$p(\lambda) = \begin{cases} \phi(\lambda) & \text{if } \lambda \in L_\kappa \\ 0 & \text{if } \lambda \in (L \cup B) - L_\kappa. \end{cases}$$

Now pick $\xi = (x_1, \ldots, x_{n+q-1})$. Observe that

$$F(l_1, l_2, \ldots, l_n, x_1, \ldots, x_{n+q}) = \text{sign}\left(\sum_{i=1}^{n} p(l_i)\right) = \phi(l_\kappa)$$

for all $(l_1, \ldots, l_n) \in L_1 \times \ldots \times L_n$, since $p(l) = 0$ for $l \notin L_\kappa$ and $p(l) = \phi(l)$ otherwise. It follows from Theorem 2 that $\text{VC}\,(\mathcal{F}_{n,q}) \geq n\lfloor\log(r)\rfloor$, as desired. ∎

## 5   The Consistency Problem

We next briefly discuss polynomial time learnability of recurrent perceptron mappings. As discussed in e.g. (Turán, 1994), in order to formalize this problem we need to first choose a *data structure* to represent the hypotheses in $\mathcal{F}_{n,q}$. In addition, since we are dealing with complexity of computation involving real numbers, we must also clarify the meaning of "finding" a hypothesis, in terms of a suitable notion of polynomial-time computation. Once this is done, the problem becomes that of solving the *consistency problem*:

> Given a set of $s \geq s(\varepsilon, \delta)$ inputs $\xi_1, \xi_2, \ldots, \xi_s \in \mathbb{R}^{n+q}$, and an arbitrary dichotomy $\Delta : \{\xi_1, \xi_2, \ldots, \xi_s\} \to \{-1, 1\}$ find a representation of a hypothesis $\phi_{\vec{c}} \in \mathcal{F}_{n,q}$ such that the restriction of $\phi_{\vec{c}}$ to the set $\{\xi_1, \xi_2, \ldots, \xi_s\}$ is identical to the dichotomy $\Delta$ (or report that no such hypothesis exists).

The representation to be used should provide an "efficient encoding" of the values of the parameters $r_1, \ldots, r_n, c_1, \ldots, c_n$: given a set of inputs $(x_1, \ldots, x_{n+q}) \in \mathbb{R}^{n+q}$, one should be able to efficiently check concept membership (that is, compute $\text{sign}\left(\sum_{i=1}^{n+q} c_i x_i\right)$). Regarding the precise meaning of polynomial-time computation, there are at least two models of complexity possible: the *unit cost model* which deals with algebraic complexity (arithmetic and comparison operations take unit time) and the *logarithmic cost model* (computation in the Turing machine sense; inputs $(x_1, \ldots, x_{n+q})$ are rationals, and the time involved in finding a representation of $r_1, \ldots, r_n, c_1, \ldots, c_n$ is required to be polynomial on the number of bits $L$.

**Theorem 3** *For each fixed $n > 0$, the consistency problem for $\mathcal{F}_{n,q}$ can be solved in time polynomial in $q$ and $s$ in the unit cost model, and time polynomial in $q$, $s$, and $L$ in the logarithmic cost model.*

Since $\text{VC}(\mathcal{F}_{n,q}) = O(n + n\log(q + 1))$, it follows from here that the class $\mathcal{F}_{n,q}$ is learnable in time polynomial in $q$ (and $L$ in the log model). Due to space limitations, we must omit the proof; it is based on the application of recent results regarding computational complexity aspects of the first-order theory of real-closed fields.

## 6   Pseudo-Dimension Bounds

In this section, we obtain results on the learnability of linear systems dynamics, that is, the class of functions obtained if one does *not* take the sign when defining recurrent perceptrons. The connection between VC dimension and sample complexity is only meaningful for classes of Boolean functions; in order to obtain learnability results applicable to real-valued functions one needs metric entropy estimates for certain spaces of functions. These can be in turn bounded through the estimation of Pollard's pseudo-dimension. We next briefly sketch the general framework for learning due to Haussler (based on previous work by Vapnik, Chervonenkis, and Pollard) and then compute a pseudo-dimension estimate for the class of interest.

The basic ingredients are two complete separable metric spaces $\mathbb{X}$ and $\mathbb{Y}$ (called respectively the sets of inputs and outputs), a class $\mathcal{F}$ of functions $f : \mathbb{X} \to \mathbb{Y}$ (called the decision rule or hypothesis space), and a function $\ell : \mathbb{Y} \times \mathbb{Y} \to [0, r] \subset \mathbb{R}$ (called the loss or cost function). The function $\ell$ is so that the class of functions $(x, y) \mapsto \ell(f(x), y)$ is "permissible" in the sense of Haussler and Pollard. Now, one may introduce, for each $f \in \mathcal{F}$, the function
$$A_{f,\ell} : \mathbb{X} \times \mathbb{Y} \times \mathbb{R} \to \{-1, 1\} : (x, y, t) \mapsto \text{sign}(\ell(f(x), y) - t)$$
as well as the class $\mathcal{A}_{\mathcal{F},\ell}$ consisting of all such $A_{f,\ell}$. The *pseudo-dimension* of $\mathcal{F}$ with respect to the loss function $\ell$, denoted by $\text{PD}[\mathcal{F}, \ell]$, is defined as:
$$\text{PD}[\mathcal{F}, \ell] := \text{VC}(\mathcal{A}_{\mathcal{F},\ell}).$$
Due to space limitations, the relationship between the pseudo-dimension and the sample complexity of the class $\mathcal{F}$ will not be discussed here; the reader is referred to the references (Haussler, 1992; Maass, 1994) for details.

For our application we define, for any two nonnegative integers $n, q$, the class
$$\mathcal{F}'_{n,q} := \left\{ \widehat{\phi_{\vec{c}}} \,\middle|\, \vec{c} \in \mathbb{R}^{n+q} \text{ is } n\text{-recursive} \right\}$$

where $\widehat{\phi_{\vec{c}}} : \mathbb{R}^{n+q} \to \mathbb{R} : (x_1, \ldots, x_{n+q}) \mapsto \sum_{i=1}^{n+q} c_i x_i$. The following Theorem can be proved using Fact 4.1.

**Theorem 4** *Let $p$ be a positive integer and assume that the loss function $\ell$ is given by $\ell(y_1, y_2) = |y_1 - y_2|^p$. Then, $\text{PD}[\mathcal{F}'_{n,q}, \ell] \leq 18n + 4n\log(p(q + 1))$.*

## Acknowledgements

This research was supported in part by US Air Force Grant AFOSR-94-0293.

## References

A.D. BACK AND A.C. TSOI, *FIR and IIR synapses, a new neural network architecture for time-series modeling*, Neural Computation, 3 (1991), pp. 375–385.

A.D. BACK AND A.C. TSOI, *A comparison of discrete-time operator models for nonlinear system identification*, Advances in Neural Information Processing Systems (NIPS'94), Morgan Kaufmann Publishers, 1995, to appear.

A.M. BAKSHO, S. DASGUPTA, J.S. GARNETT, AND C.R. JOHNSON, *On the similarity of conditions for an open-eye channel and for signed filtered error adaptive filter stability*, Proc. IEEE Conf. Decision and Control, Brighton, UK, Dec. 1991, IEEE Publications, 1991, pp. 1786–1787.

A. BLUMER, A. EHRENFEUCHT, D. HAUSSLER, AND M. WARMUTH, *Learnability and the Vapnik-Chervonenkis dimension*, J. of the ACM, 36 (1989), pp. 929-965.

D.F. DELCHAMPS, *Extracting State Information from a Quantized Output Record*, Systems and Control Letters, 13 (1989), pp. 365-372.

R.O. DUDA AND P.E. HART, *Pattern Classification and Scene Analysis*, Wiley, New York, 1973.

C.E. GILES, G.Z. SUN, H.H. CHEN, Y.C. LEE, AND D. CHEN, *Higher order recurrent networks and grammatical inference*, Advances in Neural Information Processing Systems 2, D.S. Touretzky, ed., Morgan Kaufmann, San Mateo, CA, 1990.

P. GOLDBERG AND M. JERRUM, *Bounding the Vapnik-Chervonenkis dimension of concept classes parameterized by real numbers*, Mach Learning, 18, (1995): 131-148.

D. HAUSSLER, *Decision theoretic generalizations of the PAC model for neural nets and other learning applications*, Information and Computation, 100, (1992): 78-150.

R. KOPLON AND E.D. SONTAG, *Linear systems with sign-observations*, SIAM J. Control and Optimization, 31(1993): 1245 - 1266.

W. MAASS, *Perspectives of current research about the complexity of learning in neural nets*, in *Theoretical Advances in Neural Computation and Learning*, V.P. Roychowdhury, K.Y. Siu, and A. Orlitsky, eds., Kluwer, Boston, 1994, pp. 295-336.

G.W. PULFORD, R.A. KENNEDY, AND B.D.O. ANDERSON, *Neural network structure for emulating decision feedback equalizers*, Proc. Int. Conf. Acoustics, Speech, and Signal Processing, Toronto, Canada, May 1991, pp. 1517–1520.

E.D. SONTAG, *Neural networks for control*, in *Essays on Control: Perspectives in the Theory and its Applications* (H.L. Trentelman and J.C. Willems, eds.), Birkhauser, Boston, 1993, pp. 339-380.

GYÖRGY TURÁN, *Computational Learning Theory and Neural Networks:A Survey of Selected Topics*, in *Theoretical Advances in Neural Computation and Learning*, V.P. Roychowdhury, K.Y. Siu,and A. Orlitsky, eds., Kluwer, Boston, 1994, pp. 243-293.

L.G. VALIANT *A theory of the learnable*, Comm. ACM, 27, 1984, pp. 1134–1142.

V.N.VAPNIK, *Estimation of Dependencies Based on Empirical Data*, Springer, Berlin, 1982.